# AN ANALOG VLSI CHIP FOR

# THIN-PLATE SURFACE INTERPOLATION

John G. Harris
California Institute of Technology
Computation and Neural Systems Option, 216-76
Pasadena, CA 91125

## ABSTRACT

Reconstructing a surface from sparse sensory data is a well-known problem in computer vision. This paper describes an experimental analog VLSI chip for smooth surface interpolation from sparse depth data. An eight-node 1D network was designed in $3\mu m$ CMOS and successfully tested. The network minimizes a second-order or "thin-plate" energy of the surface. The circuit directly implements the coupled depth/slope model of surface reconstruction (Harris, 1987). In addition, this chip can provide Gaussian-like smoothing of images.

## INTRODUCTION

Reconstructing a surface from sparse sensory data is a well-known problem in computer vision. Early vision modules typically supply sparse depth, orientation, and discontinuity information. The surface reconstruction module incorporates these sparse and possibly conflicting measurements of a surface into a consistent, dense depth map.

The coupled depth/slope model provides a novel solution to the surface reconstruction problem (Harris, 1987). A 1D version of this model has been implemented; fortunately, its extension to 2D is straightforward. Figure 1 depicts a high-level schematic of the circuit. The $d_i$ voltages represent noisy and possibly sparse input data, the $z_i$s are the smooth output values, and the $p_i$s are the explicitly computed slopes. The vertical data resistors (with conductance $g$) control the confidence in the input data. In the absence of data these resistors are open circuits. The horizontal chain of smoothness resistors of conductance $\lambda$ forces the derivative of the data to be smooth. This model is called the coupled depth/slope model because of the coupling between the depth and slope representations provided by the subtractor elements. The subtractors explicitly calculate a slope representation of the surface. Any depth or slope node can be made into a constraint by fixing a voltage source to the proper location in the network. Intuitively, any sudden change in slope is smoothed out with the resistor mesh.

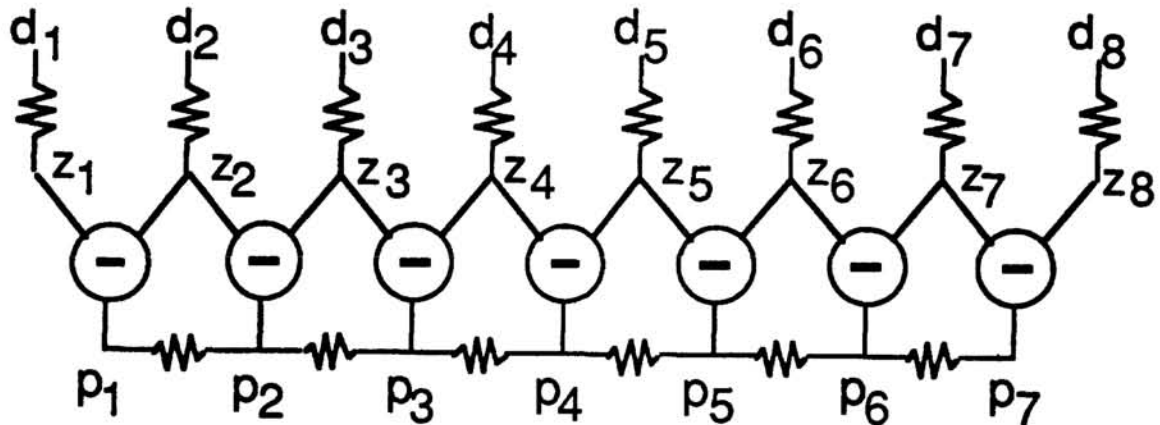

Figure 1. The coupled depth/slope model.

The tri-directional subtractor device (shown in Figure 2) is responsible for the coupling between the depth and slope representations. If nodes $A$ and $B$ are set with ideal voltage sources, then node $C$ will be forced to $A-B$ by the device. This circuit element is unusual in that all of its terminals can act as inputs or outputs. If nodes $B$ and $C$ are held constant with voltage sources, then the $A$ terminal is fixed to $B+C$. If $A$ and $C$ are input, then $B$ becomes $A-C$. When further constraints are added, this device dissipates a power proportional to $(A-B-C)^2$. In the limiting case of a continuous network, the total dissipated power is

$$E = \int \left(\frac{dz}{dx} - p\right)^2 + g\left(z - d\right)^2 + \lambda \left(\frac{dp}{dx}\right)^2 dx \qquad (1)$$

The three terms arise from the power dissipated in the subtractors and in the two different types of resistors. Energy minimization techniques and standard calculus of variations have been used to formally show that the reconstructed surfaces, $z$, satisfy the 1D biharmonic equation between input data points (Harris, 1987). In the two-dimensional formulation, $z$ is a solution of

$$\nabla^2\nabla^2 z = 0 \qquad (2)$$

This interpolant, therefore, provides the same results as minimizing the energy of a thin plate, which has been commonly used in surface reconstruction algorithms on digital computers (Grimson, 1981; Terzopoulos, 1983).

## IMPLEMENTATION

The eight-node 1D network shown in Figure 1 was designed in $3\mu$m CMOS (Mead, 1988) and fabricated through MOSIS. Three important components of the model must be mapped to analog VLSI: the two different types of resistors and the subtractors. The vertical confidence resistors are built with simple transconductance

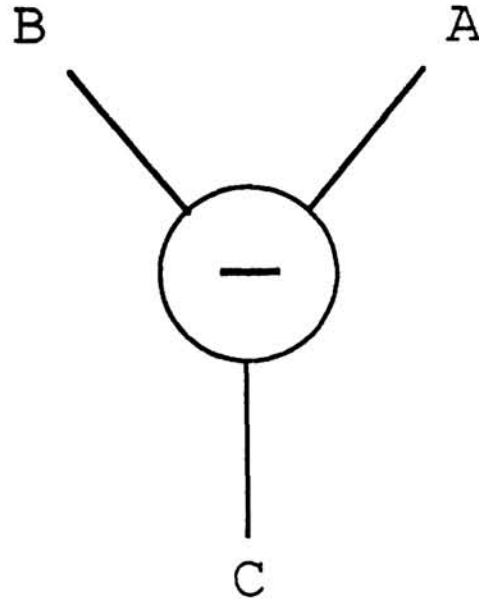

Figure 2. Tri-directional subtract constraint device.

amplifiers (transamps) connected as followers. The bias voltage of the transamp follower determines its conductance $(g)$ and therefore signifies the certainty of the data. If there are no data for a given location, the corresponding transamp follower is turned off. The horizontal smoothness resistors are implemented with Mead's saturating resistor (Mead, 1988). Since conventional CMOS processes lack adequate resistive elements, we are forced to build resistors out of transistor elements. The bias voltage for Mead's resistor allows the effective conductance of these circuit elements to vary over many orders of magnitude.

The most difficult component to implement in analog VLSI is the subtract constraint device. Its construction led to a general theory of constraint boxes which can be used to implement all sorts of constraints which are useful in early vision (Harris, 1988). The implementation of the subtract constraint device is a straightforward application of constraint box theory. Figure 3 shows a generic $n$ terminal constraint box enforcing a constraint $F$ on its voltage terminals. The constraints are enforced by generating a feedback current $I_k$ for each constrained voltage terminal. Suppose $F$ can be written as

$$F(V_0, V_1, ..., V_n) = 0 \qquad (3)$$

One possible feedback equation which implements this constraint is given by

$$I_k = -F\frac{\partial F}{\partial V_k} \qquad (4)$$

When this particular choice of feedback current is used, the constraint box minimizes the least-squares error in the constraint equation (Harris, 1989). Notice that $F$ can be scaled by any arbitrary scaling factor. This scaling factor and the capacitance at each node determine the speed of convergence of a single constraint box.

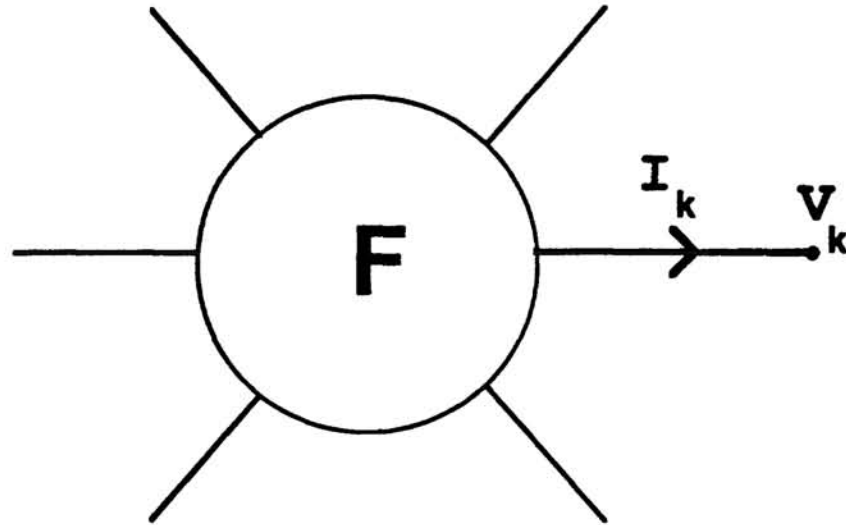

Figure 3. Generic $n$ terminal constraint box.

The subtract constraint box given in Figure 2 requires a constraint of $A - B = C$, which leads to the following error equation:

$$F(A, B, C) = A - B - C \qquad (5)$$

Straightforward application of constraint box theory yields

$$
\begin{aligned}
I_A &= -F\frac{\partial F}{\partial A} = -(A - B - C) \\[2mm]
I_B &= -F\frac{\partial F}{\partial B} = (A - B - C) \qquad (6) \\[2mm]
I_C &= -F\frac{\partial F}{\partial C} = (A - B - C)
\end{aligned}
$$

where $I_A$, $I_B$, and $I_C$ represent feedback currents that must be generated by the device.

These current feedback equations can be implemented with two modified wide-range transamps (see Figure 4). In its linear range, a single transamp produces a current proportional to the difference of its two inputs. The negative input to each transamp is indicated by an inverting circle. The transamps have been modified to produce four outputs, two positive and two negative. The negative outputs are also represented by inverting circles. Because the difference terminal $C$ can be positive or negative, it is measured with respect to a voltage reference $V_{REF}$. $V_{REF}$ is a global signal which defines zero slope. As seen in Figure 4, the

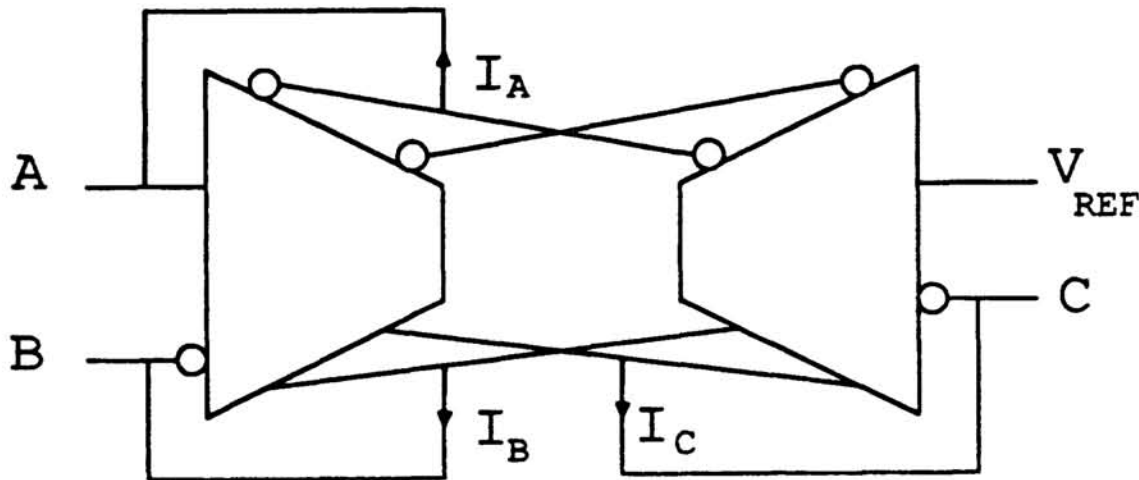

Figure 4. Tri-directional subtract constraint box.

proper combination of positive and negative outputs from the two transamps are fed back to the voltage terminals to implement the feedback equations given in eq. (6).

Analog networks which solve most regularizable early vision problems can be designed with networks consisting solely of linear resistances and batteries (Poggio and Koch, 1985). Unfortunately, many times these networks contain negative resistances that are troublesome to implement in analog VLSI. For example, the circuit shown in Figure 5 computes the same solutions as the coupled depth/slope network described in this paper. Interestingly, a 2-D implementation of this idea was implemented in the 1960s using inductors and capacitors (Volynskii and Bukhman, 1965). Proper choice of the frequency of alternating current allowed the circuit elements to act as pure positive and negative impedances. Unfortunately, negative resistances are troublesome to implement, especially in analog VLSI. One of the big advantages of using constraint boxes to implement early vision algorithms is that the resulting networks do not require negative resistances.

## ANALYSIS

Figure 6 shows a sample output of the circuit. Data (indicated by vertical dashed lines) were supplied at nodes 2, 5, and 8. As expected, the chip finds a smooth solution (solid line) which extrapolates beyond the known data points. It is well-known that a single resistive grid minimizes the first-order or membrane energy of a surface. Luo, Koch, and Mead (1988) have implemented a 48x48 resistive grid to perform surface interpolation. Figure 6 also shows the simulated performance of a first-order energy or membrane energy minimization. Data points are again supplied at nodes 2, 5, and 8. In contrast to the second-order chip results, the solution (dashed line) is much more jagged and does not extrapolate outside of

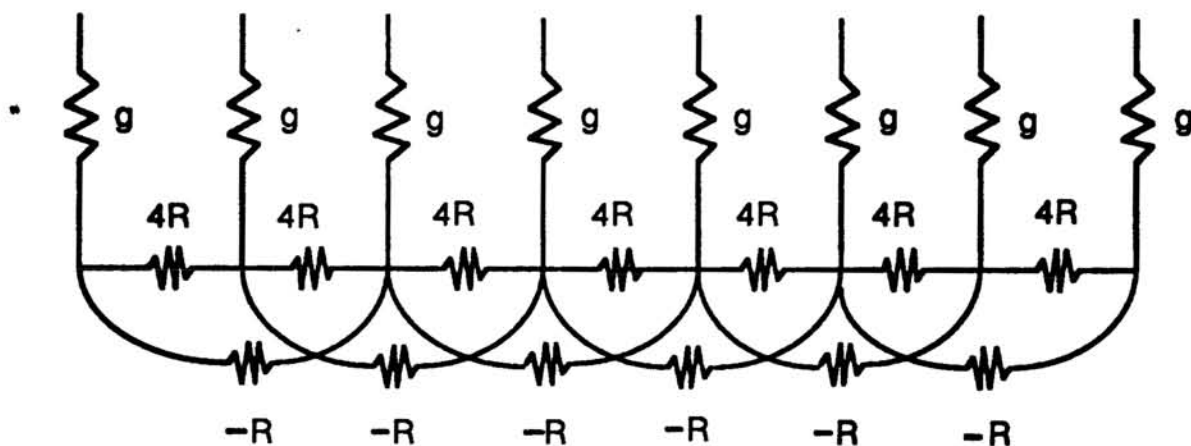

Figure 5. A negative-resistor resistor solution to the 1D biharmonic equation.

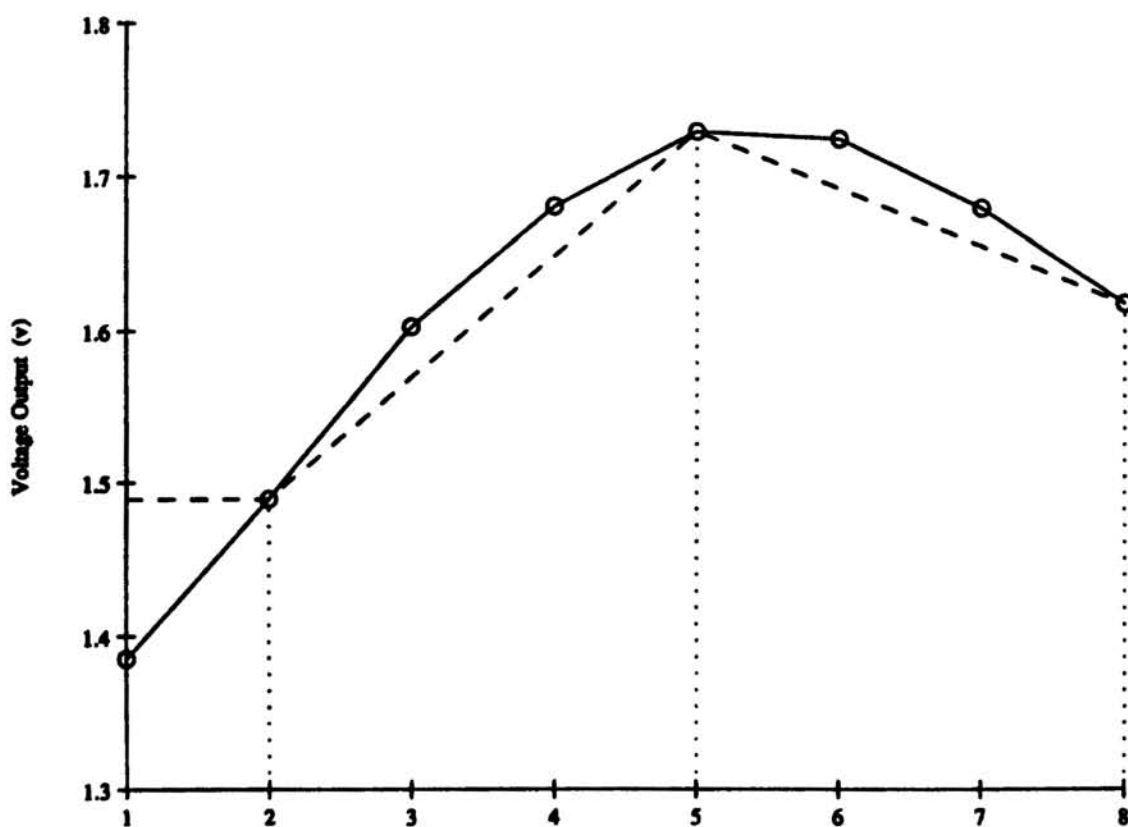

Figure 6. Measured data from the second-order chip (solid line) and simulated first-order result (dashed line).

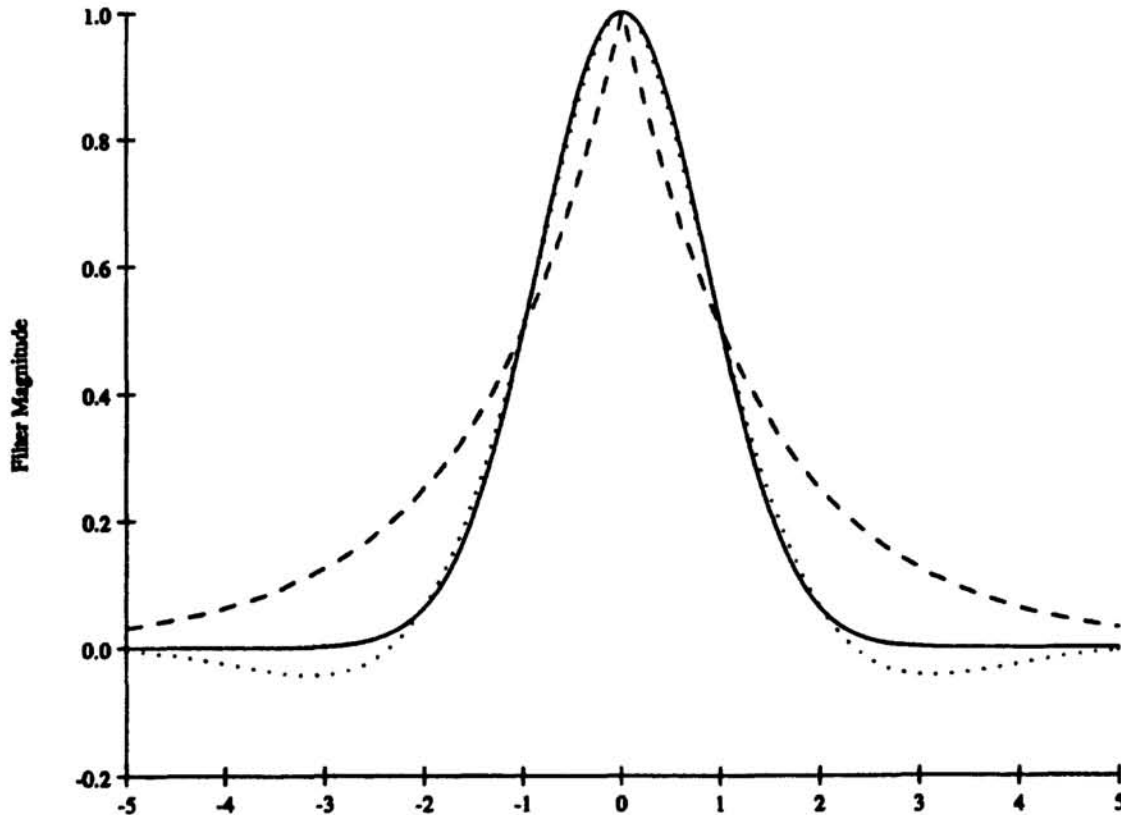

Figure 7. Graphical comparison of 1D analytic Green's functions for first-order (dashed line), second-order (dotted line) and Gaussian (solid line).

the known data points (for example, see node 1). Interestingly, psychophysics experiments support the smoother interpolant used by the second-order coupled depth/slope chip (Grimson, 1981). Unlike the second-order network, the first-order network is not rigid enough to incorporate either orientation constraints or orientation discontinuities (Terzopoulos, 1983).

Image smoothing is a special case of surface interpolation where the data are given on a dense grid. The first-order network is a poor smoothing operator. A comparison of analytic Green's function of first and second-order networks is shown in Figure 7 (the first-order shown with a dashed line and the second-order with a solid line). Note that the analytic Green's function of the second-order network (solid line) and that of standard Gaussian convolution (dotted line) are nearly identical. This fact was pointed out by Poggio, Voorhees, and Yuille (1986), when they suggested the use of the second-order energy to regularize the edge detection problem. Gaussian convolution has been claimed by many authors to be the "optimal" smoothing operator and is commonly used as the first stage of edge detection. Though the second-order network can be used to smooth images, Gaussian convolution cannot be used to solve the more difficult problem of interpolating from sparse data points.

## CONCLUSION

Biharmonic surface interpolation has been successfully demonstrated in analog VLSI. To test true performance, we plan to combine a larger version of this chip with an analog stereo network. Work has already started on building the necessary circuitry for discontinuity detection during surface reconstruction. The Gaussian-like smoothing effect of this network will be further explored through building a network with photoreceptors supplying dense data input.

### Acknowledgements

Support for this research was provided by the Office of Naval Research and the System Development Foundation. The author is a Hughes Aircraft Fellow and thanks Christof Koch and Carver Mead for their ongoing support. Additional thanks to Berthold Horn for several helpful suggestions.

### References

Grimson, W.E.L. *From Images to Surfaces*, MIT Press, Cambridge, (1981).

Harris, J.G. A new approach to surface reconstruction: the coupled depth/slope model, *Proc. IEEE First Intl. Conf. Computer Vision*, pp. 277–283, London, (1987).

Harris, J.G. Solving early vision problems with VLSI constraint networks, Neural Architectures for Computer Vision Workshop, AAAI-88, Minneapolis, Minnesota, Aug. 20 (1988).

Harris, J.G. Designing analog constraint boxes to solve energy minimization problems in vision, submitted to INNS Neural Networks Conference, Washington D.C., June (1989)

Luo, J., Koch, C., and Mead, C. An experimental subthreshold, analog CMOS two-dimensional surface interpolation circuit, Neural Information and Processing Systems Conference, Denver, Nov. (1988).

Mead, C.A. *Analog VLSI and Neural Systems*, Addison-Wesley, Reading, (1989).

Poggio, T. and Koch, C. Ill-posed problems in early vision: from computational theory to analogue networks, *Proc. R. Soc. Lond. B* 226: 303-323 (1985).

Poggio, T., Voorhees, H., and Yuille, A. A regularized solution to edge detection, *Artif. Intell. Lab Memo* No. 833, MIT, Cambridge, (1986).

Terzopoulos, D. Multilevel computational processes for visual surface reconstruction, *Comp. Vision Graph. Image Proc.* 24: 52-96 (1983).

Volynskii, B. A. and Bukhman, V. Ye. *Analogues for Solution of Boundary-Value Problems*, Pergamon Press, New York, (1965).